# A Neural Implementation of the Kalman Filter

**Robert C. Wilson**
Department of Psychology
Princeton University
Princeton, NJ 08540
`rcw2@princeton.edu`

**Leif H. Finkel**
Department of Bioengineering
University of Pennsylvania
Philadelphia, PA 19103

## Abstract

Recent experimental evidence suggests that the brain is capable of approximating Bayesian inference in the face of noisy input stimuli. Despite this progress, the neural underpinnings of this computation are still poorly understood. In this paper we focus on the Bayesian filtering of stochastic time series and introduce a novel neural network, derived from a line attractor architecture, whose dynamics map directly onto those of the Kalman filter in the limit of small prediction error. When the prediction error is large we show that the network responds robustly to changepoints in a way that is qualitatively compatible with the optimal Bayesian model. The model suggests ways in which probability distributions are encoded in the brain and makes a number of testable experimental predictions.

## 1 Introduction

There is a growing body of experimental evidence consistent with the idea that animals are somehow able to represent, manipulate and, ultimately, make decisions based on, probability distributions. While still unproven, this idea has obvious appeal to theorists as a principled way in which to understand neural computation. A key question is how such Bayesian computations could be performed by neural networks. Several authors have proposed models addressing aspects of this issue [15, 10, 9, 19, 2, 3, 16, 4, 11, 18, 17, 7, 6, 8], but as yet, there is no conclusive experimental evidence in favour of any one and the question remains open.

Here we focus on the problem of tracking a randomly moving, one-dimensional stimulus in a noisy environment. We develop a neural network whose dynamics can be shown to approximate those of a one-dimensional Kalman filter, the Bayesian model when all the distributions are Gaussian. Where the approximation breaks down, for large prediction errors, the network performs something akin to outlier or change detection and this 'failure' suggests ways in which the network can be extended to deal with more complex, non-Gaussian distributions over multiple dimensions.

Our approach rests on the modification of the line attractor network of Zhang [26]. In particular, we make three changes to Zhang's network, modifying the activation rule, the weights and the inputs in such a way that the network's dynamics map exactly onto those of the Kalman filter when the prediction error is small. Crucially, these modifications result in a network that is no longer a line attractor and thus no longer suffers from many of the limitations of these networks.

## 2 Review of the one-dimensional Kalman filter

For clarity of exposition and to define notation, we briefly review the equations behind the one-dimensional Kalman filter. In particular, we focus on tracking the true location of an object, $x(t)$, over time, $t$, based on noisy observations of its position $z(t) = x(t) + n_z(t)$, where $n_z(t)$ is zero mean Gaussian random noise with standard deviation $\sigma_z(t)$, and a model of its dynamics, $x(t+1) =$

$x(t) + v(t) + n_v(t)$, where $v(t)$ is the velocity signal and $n_v(t)$ is a Gaussian noise term with zero mean and standard deviation $\sigma_v(t)$. Assuming that $\sigma_z(t)$, $\sigma_v(t)$ and $v(t)$ are all known, then the Kalman filter's estimate of the position, $\hat{x}(t)$, can be computed via the following three equations

$$\bar{x}(t+1) = \hat{x}(t) + v(t) \tag{1}$$

$$\frac{1}{\hat{\sigma}_x(t+1)^2} = \frac{1}{\hat{\sigma}_x(t)^2 + \sigma_v(t)^2} + \frac{1}{\sigma_z(t+1)^2} \tag{2}$$

$$\hat{x}(t+1) = \bar{x}(t+1) + \frac{\hat{\sigma}_x(t+1)^2}{\sigma_z(t+1)^2} \left[ z(t+1) - \bar{x}(t+1) \right] \tag{3}$$

In equation 1 the model computes a prediction, $\bar{x}(t+1)$, for the position at time $t+1$; equation 2 updates the model's uncertainty, $\hat{\sigma}_x(t+1)$, in its estimate; and equation 3 updates the model's estimate of position, $\hat{x}(t+1)$, based on this uncertainty and the prediction error $[z(t+1) - \bar{x}(t+1)]$.

## 3 The neural network

The network is a modification of Zhang's line attractor model of head direction cells [26]. We use rate neurons and describe the state of the network at time $t$ with the membrane potential vector, $\mathbf{u}(t)$, where each component of $\mathbf{u}(t)$ denotes the membrane potential of a single neuron. In discrete time, the update equation is then

$$\mathbf{u}(t+1) = w\mathbf{J}\mathbf{f}\left[\mathbf{u}(t)\right] + \mathbf{I}(t+1) \tag{4}$$

where $w$ scales the strength of the weights, $\mathbf{J}$ is the connectivity matrix, $\mathbf{f}[\cdot]$ is the activation rule that maps membrane potential onto firing rate, and $\mathbf{I}(t+1)$ is the input at time $t+1$. As in [26], we set $\mathbf{J} = \mathbf{J}^{sym} + \gamma(t)\mathbf{J}^{asym}$ such that the the connections are made up of a mixture of symmetric, $\mathbf{J}^{sym}$, and asymmetric components, $\mathbf{J}^{asym}$ (defined as spatial derivative of $\mathbf{J}^{sym}$), with mixing strength $\gamma(t)$ that can vary over time. Although the results presented here do not depend strongly on the exact forms of $\mathbf{J}^{sym}$ and $\mathbf{J}^{asym}$, for concreteness we use the following expressions

$$J_{ij}^{sym} = K_w \exp\left[ \frac{\cos\left(\frac{2\pi(i-j)}{N}\right) - 1}{\sigma_w^2} \right] - c \quad ; \quad J_{ij}^{asym} = -\frac{2\pi}{N\sigma_w^2}\sin\left(\frac{2\pi(i-j)}{N}\right)J_{ij}^{sym} \tag{5}$$

where $N$ is the number of neurons in the network and $\sigma_w$, $K_w$ and $c$ are constants that determine the width and excitatory and inhibitory connection strengths respectively.

To approximate the Kalman filter, the activation function must implement divisive inhibition [14, 13]

$$\mathbf{f}[\mathbf{u}] = \frac{[\mathbf{u}]_+}{S + \mu\sum_i[u_i]_+} \tag{6}$$

where $[\mathbf{u}]_+$ denotes recitification of $\mathbf{u}$; $\mu$ determines the strength of the divisive feedback and $S$ determines the gain when there is no previous activity in the network.

When $w = 1$, $\gamma(t) = 0$ and $\mathbf{I}(t) = 0$, the network is a line attractor over a wide range of $K_w$, $\sigma_w$, $c$, $S$ and $\mu$, having a continuum of fixed points (as $N \to \infty$). Each fixed point has the same shape, taking the form of a smooth membrane potential profile, $\mathbf{U}(x) = \mathbf{J}^{sym}\mathbf{f}\left[\mathbf{U}(x)\right]$, centered at location, $x$, in the network.

When $\gamma(t) \neq 0$, the bump of activity can be made to move over time (without losing its shape) [26] and hence, so long as $\gamma(t) = v(t)$, implement the prediction step of the Kalman filter (equation 1). That is, if the bump at time $t$ is centered at $\hat{x}(t)$, i.e. $\mathbf{u}(t) = \mathbf{U}(\hat{x}(t))$, then at time $t+1$ it is centered at $\bar{x}(t+1) = \hat{x}(t) + \gamma(t)$, i.e. $\mathbf{u}(t+1) = \mathbf{U}(\hat{x}(t) + \gamma(t)) = \mathbf{U}(\bar{x}(t+1))$. Thus, in this configuration, the network can already implement the first step of the Kalman filter through its recurrent connectivity. The next two steps, equations 2 and 3, however, remain inaccessible as the network has no way of encoding uncertainty and it is unclear how it will deal with external inputs.

## 4 Relation to Kalman filter - small prediction error case

In this section we outline how the neural network dynamics can be mapped onto those of a Kalman filter. In the interests of space we focus only on the main points of the derivation, leaving the full working to the supplementary material.

Our approach is to analyze the network in terms of $\mathbf{U}$, which, for clarity, we define here to be the fixed point membrane potential profile of the network when $w = 1$, $\gamma(t) = 0$, $\mathbf{I}(t) = 0$, $S = S_0$ and $\mu = \mu_0$. Thus, the results described here are independent of the exact form of $\mathbf{U}$ so long as it is a smooth, non-uniform profile over the network.

We begin by making the assumption that both the input, $\mathbf{I}(t)$, and the network membrane potential, $\mathbf{u}(t)$, take the form of scaled versions $\mathbf{U}$, with the former encoding the noisy observations, $z(t)$, and the latter encoding the network's estimate of position, $\hat{x}(t)$, i.e.,

$$\mathbf{I}(t) = A(t)\mathbf{U}(z(t)) \quad \text{and} \quad \mathbf{u}(t) = \alpha(t)\mathbf{U}(\hat{x}(t)) \tag{7}$$

Substituting this *ansatz* for membrane potential into the left hand side of equation 4 gives

$$LHS = \alpha(t+1)\mathbf{U}(\hat{x}(t+1)) \tag{8}$$

and into the right hand side of equation 4 gives

$$RHS = \underbrace{w\mathbf{Jf}\left[\alpha(t)\mathbf{U}(\hat{x}(t))\right]}_{\text{recurrent input}} + \underbrace{A(t+1)\mathbf{U}(z(t+1))}_{\text{external input}} \tag{9}$$

For the *ansatz* to be self-consistent we require that $RHS$ can be written in the same form as $LHS$. We now show that this is the case.

As in the previous section, the recurrent input, implements the prediction step of the Kalman filter, which, after a little algebra (see supplementary material), allows us to write

$$RHS \approx \underbrace{C\mathbf{U}(\bar{x}(t+1))}_{\text{prediction}} + \underbrace{A(t+1)\mathbf{U}(z(t+1))}_{\text{external input}} \tag{10}$$

With the variable $C$ defined as

$$C = \frac{1}{\frac{S}{w(S_0+\mu_0\mathcal{I})}\frac{1}{\alpha(t)} + \frac{\mu\mathcal{I}}{w(S_0+\mu_0\mathcal{I})}} \tag{11}$$

where $\mathcal{I} = \sum_i [U_i(\hat{x}(t))]_+$.

If we now suppose that the prediction error $[z(t+1) - \bar{x}(t+1)]$ is small, then we can linearize around the prediction, $\bar{x}(t+1)$, to get (see supplementary material)

$$RHS \approx [C + A(t+1)]\mathbf{U}\left(\bar{x}(t+1) + \frac{A(t+1)}{A(t+1)+C}[z(t+1) - \bar{x}(t+1)]\right) \tag{12}$$

which is of the same form as equation 8 and thus the *ansatz* holds. More specifically, equating terms in equations 8 and 12, we can write down expressions for $\alpha(t+1)$ and $\hat{x}(t+1)$

$$\alpha(t+1) \approx C + A(t+1) = \frac{1}{\frac{S}{w(S_0+\mu_0\mathcal{I})}\frac{1}{\alpha(t)} + \frac{\mu\mathcal{I}}{w(S_0+\mu_0\mathcal{I})}} + A(t+1) \tag{13}$$

$$\hat{x}(t+1) \approx \bar{x}(t) + \frac{A(t+1)}{\alpha(t+1)}[z(t+1) - x(t+1)] \tag{14}$$

which, if we define $w$ such that

$$\frac{S}{w(S_0+\mu_0\mathcal{I})} = 1 \quad \text{i.e.} \quad w = \frac{S}{S_0+\mu_0\mathcal{I}} \tag{15}$$

are identical to equations 2 and 3 so long as

$$\text{(a)} \quad \alpha(t) \propto \frac{1}{\hat{\sigma}_x(t)^2} \qquad \text{(b)} \quad A(t) \propto \frac{1}{\sigma_z(t)^2} \qquad \text{(c)} \quad \frac{\mu\mathcal{I}}{S} \propto \sigma_v(t)^2 \tag{16}$$

Thus the network dynamics, when the prediction error is small, map directly onto the Kalman filter equations. This is our main result.

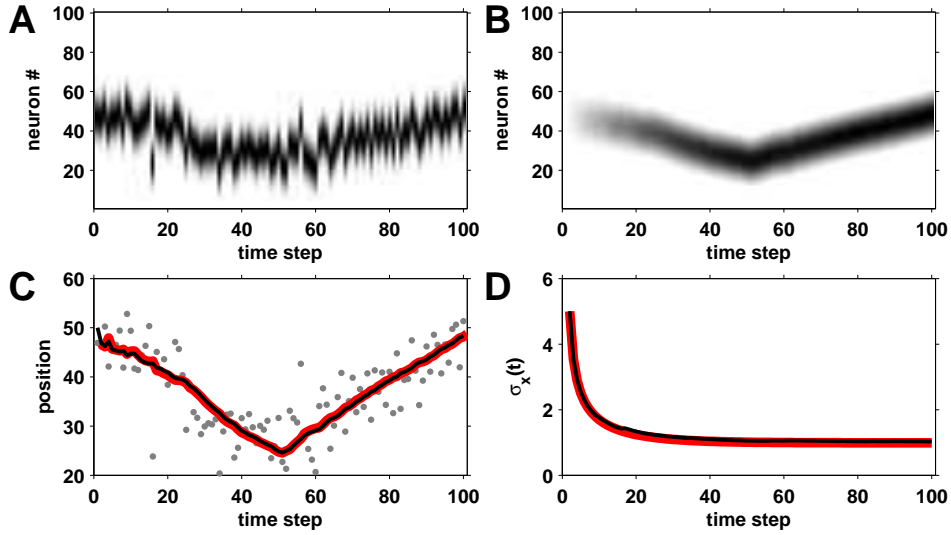

Figure 1: Comparison of noiseless network dynamics with dynamics of the Kalman Filter for small prediction errors.

## 4.1 Implications

**Reciprocal code for uncertainty in input and estimate**  Equation 16a provides a link between the strength of activity in the network and the overall uncertainty in the estimate of the Kalman filter, $\hat{\sigma}_x(t)$, with uncertainty decreasing as the activity increases. A similar relation is also implied for the uncertainty in the observations, $\sigma_z(t)$, where equation 16b suggests that this should be reflected in the magnitude of the input, $A(t)$. Interestingly, such a scaling, *without* a corresponding narrowing of tuning curves, is seen in the brain [20, 5, 2].

**Code for velocity signal**  As with Zhang's line attractor network [26], the mean of the velocity signal, $v(t)$ is encoded into the recurrent connections of the network, with the degree of asymmetry in the weights, $\gamma(t)$, proportional to the speed. Such hard coding of the velocity signal represents a limitation of the model, as we would like to be able to deal with arbitrary, time varying speeds. However, this kind of change could be implemented by pre-synaptic inhibition [24] or by using a 'double-ring' network similar to [25].

Equation 16c implies that the variance of the velocity signal, $\sigma_v(t)$, is encoded in the strength of the divisive feedback, $\mu$ (assuming constant $S$). This is very different from Zhang's model, that has no concept of uncertainty and is also very different from the traditional view of divisive inhibition that sees it as a mechanism for gain control [14, 13].

**The network is no longer a line attractor**  This can be seen by considering the fixed point values of the scale factor, $\alpha(t)$, when the input current, $\mathbf{I}(t) = 0$. Requiring $\alpha(t+1) = \alpha(t) = \alpha^*$ in equation 13 gives values for these fixed points as

$$\alpha^* = 0 \qquad \text{and} \qquad \alpha^* = \left(\frac{S_0 + \mu_0 \mathcal{I}}{\mu \mathcal{I}}\right) w - \frac{S}{\mu \mathcal{I}} \tag{17}$$

This second solution is exactly zero when $w$ satisfies equation 15, hence the network only has one fixed point corresponding to the all zero state and is *not* a line attractor. This is a key result as it removes all of the constraints required for line attractor dynamics such as infinite precision in the weights and lack of noise in the network and thus the network is much more biologically plausible.

## 4.2 An example

In figure 1 we demonstrate the ability of the network to approximate the dynamics of a one-dimensional Kalman filter. The input, shown in figure 1A, is a noiseless bump of current centered

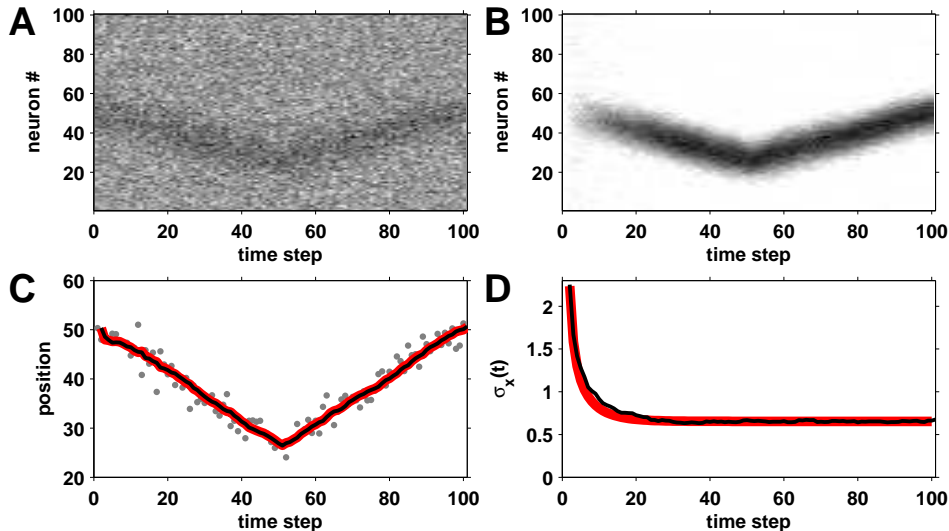

Figure 2: Response of the network when presented with a noisy moving bump input.

at the position of the observation, $z(t)$. The observation noise has standard deviation $\sigma_z(t) = 5$, the speed $v(t) = 0.5$ for $1 \leq t < 50$ and $v(t) = -0.5$ for $50 \leq t < 100$ and the standard deviation of the random walk dynamics, $\sigma_v(t) = 0.2$. In accordance with equation 16b, the height of each bump is scaled by $1/\sigma_z(t)^2$.

In figure 1B we plot the output activity of the network over time. Darker shades correspond to higher firing rates. We assume that the network gets the correct velocity signal, i.e. $\gamma(t) = v(t)$ and $\mu$ is set such that equation 16c holds. The other parameters are set to $K_w = 1$, $\sigma_w = 0.2$, $c = 0.05$, $S = S_0 = 1$ and $\mu_0 = 1$ which gives $\mathcal{I} = 5.47$. As can be seen from the plot, the amount of activity in the network steadily grows from zero over time to an asymptotic value, corresponding to the network's increasing certainty in its predictions. The position of the bump of activity in the network is also much less jittery than the input bump, reflecting a certain amount of smoothing.

In figure 1C we compare the positions of the input bumps (gray dots) with the position of the network bump (black line) and the output of the equivalent Kalman filter (red line). The network clearly tracks the Kalman filter estimate extremely well. The same is true for the network's estimate of the uncertainty, computed as $1/\sqrt{\alpha(t)}$ and shown as the black line in figure 1D, which tracks the Kalman filter uncertainty (red line) almost exactly.

## 5   Effect of input noise

We now consider the effect of noise on the ability of the network to implement a Kalman filter. In particular we consider noise in the input signal, which for this simple one layer network is equivalent to having noise in the update equation. For brevity, we only present the main results along with the results of simulations, leaving more detailed analysis to the supplementary material.

Specifically, we consider input signals where the only source of noise is in the input current i.e. there is no additional jitter in the position of the bump as there was in the noiseless case, thus we write

$$\mathbf{I}(t) = A(t)\mathbf{U}\left(x(t)\right) + \epsilon(t) \tag{18}$$

where $\epsilon(t)$ is some noise vector. The main effect of the noise is that it perturbs the effective position of the input bump. This can be modeled by extracting the maximum likelihood estimate of the input position given the noisy input and then using this position as the input to the equivalent Kalman filter. Because of the noise, this extracted position is not, in general, the same as the noiseless input position and for zero mean Gaussian noise with covariance $\Sigma$, the variance of the perturbation, $\sigma_z(t)$,

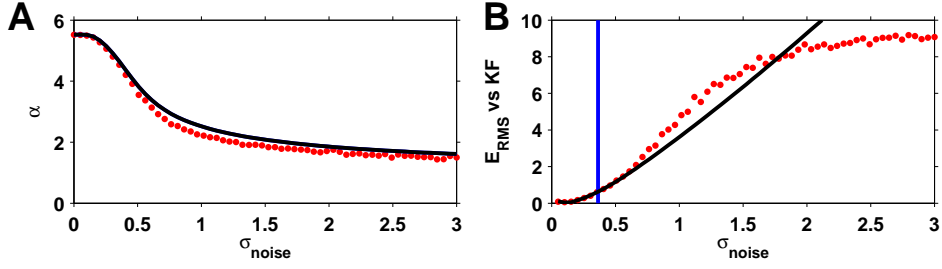

Figure 3: Effect of noise magnitude on performance of network.

is approximately given by

$$\sigma_z(t) \approx \frac{1}{A(t)} \sqrt{\frac{2}{\mathbf{U}'^T \Sigma^{-1} \mathbf{U}'}} \tag{19}$$

Now, for the network to approximate a Kalman filter, equation 16b must hold which means that we require the magnitude of the covariance matrix to scale in proportion to the strength of the input signal, $A(t)$, i.e. $\Sigma \propto A(t)$. Interestingly this relation is true for Poisson noise, the type of noise that is found all over the brain.

In figure 2 we demonstrate the ability of the network to approximate a Kalman filter. In panel A we show the input current which is a moving bump of activity corrupted by independent Gaussian noise of standard deviation $\sigma_{noise} = 0.23$, or about two thirds of the maximum height of the fixed point bump, $\mathbf{U}$. This is a high noise setting and it is hard to see the bump location by eye. The network dramatically cleans up this input signal (figure 2B) and the output activity, although still noisy, reflects the position of the underlying stimulus much more faithfully than the input. (Note that the colour scales in A and B are different).

In panel C we compare the position of the output bump in the network (black line) with that of the equivalent Kalman filter. To do this we first fit the noisy input bump at each time to obtain input positions $z(t)$ shown as gray dots. Then using $\sigma_z = 2.23$ computed using equation 19 we can compute the estimates of the equivalent Kalman filter (thick red line). which closely match those of the network (black line). Similarly, there is good agreement between the two estimates of the uncertainty, $\hat{\sigma}_x(t)$, panel D (black line - network, red line - Kalman filter).

### 5.1 Performance of the network as a function of noise magnitude

The noise not only affects the position of the input bump but also, in a slightly more subtle manner, causes a gradual decline in the ability of the network to emulate a Kalman filter. The reason for this (outlined in more detail in the supplementary material) is that the output bump scale factor, $\alpha$, decreases as a function of the noise variance, $\sigma_{noise}$. This effect is illustrated in figure 3A where we plot the steady state value of $\alpha$ (for constant input strength, $A(t)$) as a function of $\sigma_{noise}$. The average results of simulations on 100 neurons are shown as the red dots, while the black line represents the results of the theory in the supplementary material.

The reason for the decline in $\alpha$ as $\sigma_{noise}$ goes up is that, because of the rectifying non-linearity in the activation rule, increasing $\sigma_{noise}$ increases the amount of noisy activity in the network. Because of inhibition (both divisive and subtractive) in the network, this 'noisy activity' competes with the bump activity and decreases it - thus reducing $\alpha$.

This decrease in $\alpha$ results in a change in the Kalman gain of the network, by equation 14, making it different from that of the equivalent Kalman filter, thus degrading the network's performance. We quantify this difference in figure 3B where we plot the root mean squared error (in units of neural position) between the network and the equivalent Kalman filter as a function of $\sigma_{noise}$. As before, the results of simulations are shown as red dots and the theory (outlined in the supplementary material) is the black line. To give some sense for the scale on this plot, the horizontal blue line corresponds to the maximum height of the (noise free) input bump. Thus we may conclude that the performance of the network and the theory are robust up to fairly large values of $\sigma_{noise}$.

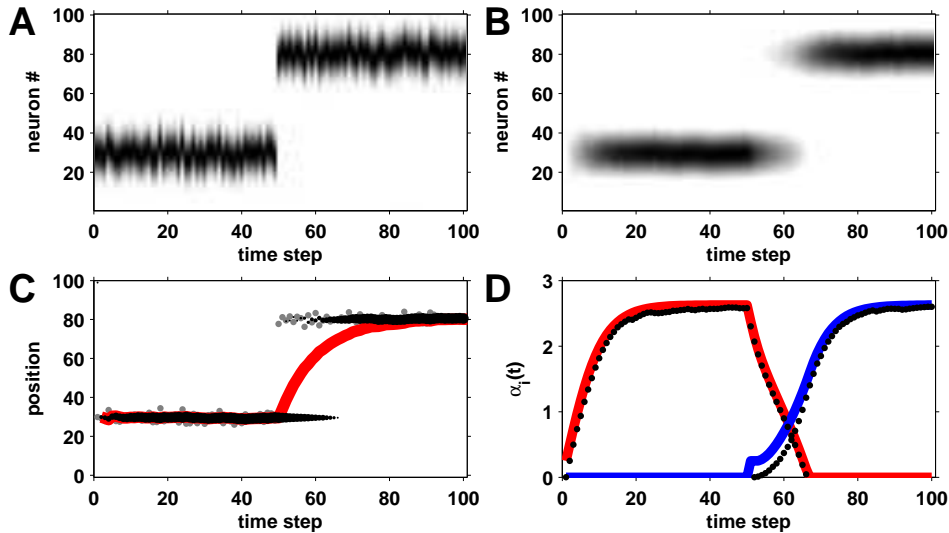

Figure 4: Response of the network to changepoints.

## 6 Response to changepoints (and outliers) - large prediction error case

We now consider the dynamics of the network when the prediction error is large. By large we mean that the prediction error is greater than the width of the bump of activity in the network. Such a big discrepancy could be caused by an outlier or a changepoint, i.e. a sustained large and abrupt change in the input position at a random time. In the interests of space we focus only on the latter case and such an input, with a changepoint at $t = 50$, is shown in figure 4A.

In figure 4B we show the network's response to this stimulus. As before, prior to the change, there is a single bump of activity whose position approximates that of a Kalman filter. However, after the changepoint, the network maintains two bumps of activity for several time steps. One at the original position, that shrinks over time and essentially predicts where the input would be if the change had not occurred, and a second, that grows over time, at the location of the input after the changepoint. Thus in the period immediately after the changepoint, the network can be thought of as encoding two separate and competing hypotheses about the position of the stimulus, one corresponding to the case where no change has occurred, and the other, the case where a change occurred at $t = 50$.

In figure 4C we compare the position of the bump(s) in the network (black dots whose size reflects the size of each bump) to the output from the Kalman filter (red line). Before the changepoint, the two agree well, but after the change, the Kalman filter becomes suboptimal, taking a long time to move to the new position. The network, however, by maintaining two hypotheses reacts much better.

Finally, in figure 4D we plot the scale factor, $\alpha_i(t)$, of each bump as computed from the simulations (black dots) and from the approximate analytic solution described in the supplementary material (red line for bump at 30, blue line for bump at 80). As can be seen, there is good agreement between theory and simulation, with the largest discrepancy occurring for small values of the scale factor.

Thus, when confronted with a changepoint, the network no longer approximates a Kalman filter and instead maintains two competing hypotheses in a way that is qualitatively similar to that of the run-length distribution in [1]. This is an extremely interesting result and hints at ways in which more complex distributions may be encoded in these type of networks.

## 7 Discussion

### 7.1 Relation to previous work

Of the papers mentioned in the introduction, two are of particular relevance to the current work. In the first, [8], the authors considered a neural implementation of the Kalman filter using line

attractors. Although this work, at first glance, seems similar to what is presented here, there are several major differences, the main one being that our network is not a line attractor at all, while the results in [8] rely on this property. Also, in [8], the Kalman gain is changed manually, where as in our case it adjusts automatically (equations 13 and 14), and the form of non-linearity is different.

Probabilistic population coding [16, 4] is more closely related to model presented here. Combined with divisive normalization, these networks can implement a Kalman filter exactly, while the model presented here can 'only' approximate one. While this may seem like a limitation of our network, we see it as an advantage as the breakdown of the approximation leads to a more robust response to outliers and changepoints than a pure Kalman filter.

## 7.2   Extension beyond one-dimensional Gaussians

A major limitation of the current model is that it only applies to one-dimensional Gaussian tracking - clearly an unreasonable restriction for the brain. One possible way around this limitation is hinted at by the response of the network in the changepoint case where we saw two, largely independent bumps of activity in the network. This ability to encode multiple 'particles' in the network may allow networks of this kind to implement something like the dynamics of a particle filter [12] that can approximate the inference process for non-linear and non-Gaussian systems. Such a possibility is an intriguing idea for future work.

## 7.3   Experimental predictions

The model makes at least two easily testable predictions about the response of head direction cells [21, 22, 23] in rats. The first comes by considering the response of the neurons in the 'dark'. Assuming that all bearing cues can indeed be eliminated, by setting $A(t) = 0$ in equation 13, we expect the activity of the neurons to fall off as $1/t$ and that the shape of the tuning curves will remain approximately constant. Note that this prediction is vastly different from the behaviour of a line attractor, where we would not expect the level of activity to fall off at all in the dark.

Another, slightly more ambitious experiment would involve perturbing the reliability of one of the landmark cues. In particular, one could imagine a training phase, where the position of one landmark is jittered over time, such that each time the rat encounters it it is at a slightly different heading. In the test case, all other, reliable, landmark cues would be removed and the response of head direction cells measured in response to presentation of the unreliable cue alone. The prediction of the model is that this would reduce the strength of the input, $A$, which in turn reduces the level of activity in the head direction cells, $\alpha$. In particular, if $\sigma_z$ is the jitter of the unreliable landmark, then we expect $\alpha$ to scale as $1/\sigma_z^2$. This prediction is very different from that of a line attractor which would predict a constant level of activity regardless of the reliability of the landmark cues.

# 8   Conclusions

In this paper we have introduced a novel neural network model whose dynamics map directly onto those of a one-dimensional Kalman filter when the prediction error is small. This property is robust to noise and when the prediction error is large, such as for changepoints, the output of the network diverges from that of the Kalman filter, but in a way that is both interesting and useful. Finally, the model makes two easily testable experimental predictions about head direction cells.

**Acknowledgements**

We would like to thank the anonymous reviewers for their very helpful comments on this work.

# References

[1] R.P. Adams and D.J.C. MacKay. Bayesian online changepoint detection. Technical report, University of Cambridge, Cambridge, UK, 2007.

[2] J. S. Anderson, I. Lampl, D. C. Gillespie, and D. Ferster. The contribution of noise to contrast invariance of orientation tuning in cat visual cortex. *Science*, 290:1968–1972, 2000.

[3] M. J. Barber, J. W. Clark, and C. H. Anderson. Neural representation of probabilistic information. *Neural Computation*, 15:1843–1864, 2003.

[4] J. Beck, W. J. Ma, P. E. Latham, and A. Pouget. Probabilistic population codes and the exponential family of distributions. *Progress in Brain Research*, 165:509–519, 2007.

[5] K. H. Britten, M. N. Shadlen, W. T. Newsome, and J. A. Movshon. Response of neurons in macaque mt to stochastic motion signals. *Visual Neuroscience*, 10(1157-1169), 1993.

[6] S. Deneve. Bayesian spiking neurons i: Inference. *Neural Computation*, 20:91–117, 2008.

[7] S. Deneve. Bayesian spiking neurons ii: Learning. *Neural Computation*, 20:118–145, 2008.

[8] S. Deneve, J.-R. Duhammel, and A. Pouget. Optimal sensorimotor integration in recurrent cortical networks: a neural implementation of kalman filters. *Journal of Neuroscience*, 27(21):5744–5756, 2007.

[9] S. Deneve, P. E. Latham, and A. Pouget. Reading population codes: a neural implementation of ideal observers. *Nature Neuroscience*, 2(8):740–745, 1999.

[10] S. Deneve, P. E. Latham, and A. Pouget. Efficient computation and cue integration with noisy population codes. *Nature Neuroscience*, 4(8):826–831, 2001.

[11] J. I. Gold and M. N. Shadlen. Representation of a perceptual decision in developing oculomotor commands. *Nature*, 404(390-394), 2000.

[12] N. J. Gordon, D. J. Salmond, and A. F. M. Smith. Novel approach to nonlinear/non-gaussian bayesian state estimation. *IEE-Proceedings-F*, 140:107–113, 1993.

[13] D. J. Heeger. Modeling simple cell direction selectivity with normalized half-squared, linear operators. *Journal of Neurophysiology*, 70:1885–1897, 1993.

[14] D. J. Heeger. Normalization of cell responses in cat striate cortex. *Visual Neuroscience*, 9:181–198, 1993.

[15] P. E. Latham, S. Deneve, and A. Pouget. Optimal computation with attractor networks. *Journal of Physiology Paris*, 97(683-694), 2003.

[16] W. J. Ma, J. M. Beck, P. E. Latham, and A. Pouget. Bayesian inference with probabilistic population codes. *Nature Neuroscience*, 9(11):1432–1438, 2006.

[17] R. P. N. Rao. Bayesian computation in recurrent neural circuits. *Neural Computation*, 16:1–38, 2004.

[18] R. P. N. Rao. Hierarchical bayesian inference in networks of spiking neurons. In *Advances in Neural Information Processing Systems*, volume 17, 2005.

[19] M. Sahani and P. Dayan. Doubly distributional population codes: simultaneous representation of uncertainty and multiplicity. *Neural Computation*, 15:2255–2279, 2003.

[20] G. Sclar and R. D. Freeman. Orientation selectivity in the cat's striate cortex is invariant with stimulus contrast. *Experimental Brain Research*, 46:457–461, 1982.

[21] J. S. Taube, R. U. Muller, and J. B. Ranck. Head-direction cells recorded from postsubiculum in freely moving rats. i. description and quantitative analysis. *Journal of Neuroscience*, 10(2):420–435, 1990.

[22] J. S. Taube, R. U. Muller, and J. B. Ranck. Head-direction cells recorded from postsubiculum in freely moving rats. ii. effects of environmental manipulations. *Journal of Neuroscience*, 10(2):436–447, 1990.

[23] S. I. Wiener and J. S. Taube. *Head direction cells and the neural mechanisms of spatial orientation*. MIT Press, 2005.

[24] L.-G. Wu and P. Saggau. Presynaptic inhibition of elicited neurotransmitter release. *Trends in Neuroscience*, 20:204–212, 1997.

[25] X. Xie, R. H. Hahnloser, and H. S. Seung. Double-ring network model of the head-direction system. *Physical Review E*, 66:0419021–0419029, 2002.

[26] K. Zhang. Representation of spatial orientation by the intrinsic dynamics of the head-direction cell ensemble: a theory. *Journal of Neuroscience*, 16(6):2112–2126, 1996.

